# Shape Context: A new descriptor for shape matching and object recognition

**Serge Belongie, Jitendra Malik and Jan Puzicha**
Department of Electrical Engineering and Computer Sciences
University of California at Berkeley
Berkeley, CA 94720, USA
{*sjb,malik,puzicha*} *@cs.berkeley.edu*

## Abstract

We develop an approach to object recognition based on matching shapes and using a resulting measure of similarity in a nearest neighbor classifier. The key algorithmic problem here is that of finding pointwise correspondences between an image shape and a stored prototype shape. We introduce a new shape descriptor, the *shape context*, which makes this possible, using a simple and robust algorithm. The shape context at a point captures the distribution over relative positions of other shape points and thus summarizes global shape in a rich, local descriptor. We demonstrate that shape contexts greatly simplify recovery of correspondences between points of two given shapes. Once shapes are aligned, shape contexts are used to define a robust score for measuring shape similarity. We have used this score in a nearest-neighbor classifier for recognition of hand written digits as well as 3D objects, using exactly the same distance function. On the benchmark MNIST dataset of handwritten digits, this yields an error rate of 0.63%, outperforming other published techniques.

## 1    Introduction

The last decade has seen increased application of statistical pattern recognition techniques to the problem of object recognition from images. Typically, an image block with $n$ pixels is regarded as an $n$ dimensional feature vector formed by concatenating the brightness values of the pixels. Given this representation, a number of different strategies have been tried, e.g. nearest-neighbor techniques after extracting principal components [15, 13], convolutional neural networks [12], and support vector machines [14, 5]. Impressive performance has been demonstrated on datasets such as digits and faces.

A vector of pixel brightness values is a somewhat unsatisfactory representation of an object. Basic invariances e.g. to translation, scale and small amount of rotation must be obtained by suitable pre-processing or by the use of enormous amounts of training data [12]. Instead, we will try to extract "shape", which by definition is required to be invariant under a group of transformations. The problem then becomes that of

operationalizing a definition of shape. The literature in computer vision and pattern recognition is full of definitions of shape descriptors and distance measures, ranging from moments and Fourier descriptors to the Hausdorff distance and the medial axis transform. (For a recent overview, see [16].) Most of these approaches suffer from one of two difficulties: (1) Mapping the shape to a small number of numbers, e.g. moments, loses information. Inevitably, this means sacrificing discriminative power. (2) Descriptors restricted to silhouettes and closed curves are of limited applicability. Shape is a much more general concept.

Fundamentally, shape is about relative positional information. This has motivated approaches such as [1] who find key points or landmarks, and recognize objects using the spatial arrangements of point sets. However not all objects have distinguished key points (think of a circle for instance), and using key points alone sacrifices the shape information available in smooth portions of object contours.

Our approach therefore uses a general representation of shape – a set of points sampled from the contours on the object. Each point is associated with a novel descriptor, the *shape context*, which describes the coarse arrangement of the rest of the shape with respect to the point. This descriptor will be different for different points on a single shape $S$; however corresponding (homologous) points on similar shapes $S$ and $S'$ will tend to have similar shape contexts. Correspondences between the point sets of $S$ and $S'$ can be found by solving a bipartite weighted graph matching problem with edge weights $C_{ij}$ defined by the similarity of the shape contexts of points $i$ and $j$. Given correspondences, we can effectively calculate the similarity between the shapes $S$ and $S'$. This similarity measure is then employed in a nearest-neighbor classifier for object recognition.

The core of our work is the concept of shape contexts and its use for solving the correspondence problem between two shapes. It can be compared to an alternative framework for matching point sets due to Gold, Rangarajan and collaborators (e.g. [7, 6]). They propose an iterative optimization algorithm to jointly determine point correspondences and underlying image transformations. The cost measure is Euclidean distance between the first point set and a transformed version of the second point set. This formulation leads to a difficult non-convex optimization problem which is solved using deterministic annealing. Another related approach is elastic graph matching [11] which also leads to a difficult stochastic optimization problem.

## 2  Matching with Shape Contexts

In our approach, a shape is represented by a discrete set of points sampled from the internal or external contours on the shape. These can be obtained as locations of edge pixels as found by an edge detector, giving us a set $\mathcal{P} = \{p_1, \ldots, p_n\}$, $p_i \in \mathbb{R}^2$, of $n$ points. They need not, and typically will not, correspond to key-points such as maxima of curvature or inflection points. We prefer to sample the shape with roughly uniform spacing, though this is also not critical. Fig. 1(a,b) shows sample points for two shapes. For each point $p_i$ on the first shape, we want to find the "best" matching point $q_j$ on the second shape. This is a correspondence problem similar to that in stereopsis. Experience there suggests that matching is easier if one uses a rich local descriptor instead of just the brightness at a single pixel or edge location. Rich descriptors reduce the ambiguity in matching.

In this paper, we propose a descriptor, the *shape context*, that could play such a role in shape matching. Consider the set of vectors originating from a point to all other sample points on a shape. These vectors express the configuration of the entire shape relative to the reference point. Obviously, this set of $n-1$ vectors is a rich

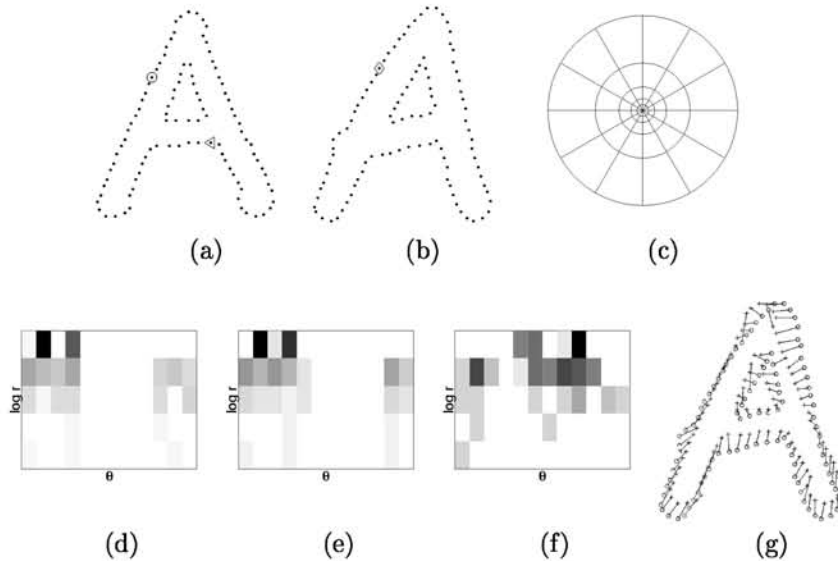

(a)      (b)      (c)

(d)      (e)      (f)      (g)

Figure 1: Shape context computation and matching. (a,b) Sampled edge points of two shapes. (c) Diagram of log-polar histogram bins used in computing the shape contexts. We use 5 bins for $\log r$ and 12 bins for $\theta$. (d-f) Example shape contexts for reference samples marked by ○, ◇, ◁ in (a,b). Each shape context is a log-polar histogram of the coordinates of the rest of the point set measured using the reference point as the origin. (Dark=large value.) Note the visual similarity of the shape contexts for ○ and ◇, which were computed for relatively similar points on the two shapes. By contrast, the shape context for ◁ is quite different. (g) Correspondences found using bipartite matching, with costs defined by the $\chi^2$ distance between histograms.

description, since as $n$ gets large, the representation of the shape becomes exact. The full set of vectors as a shape descriptor is much too detailed since shapes and their sampled representation may vary from one instance to another in a category. We identify the *distribution* over relative positions as a more robust and compact, yet highly discriminative descriptor. For a point $p_i$ on the shape, we compute a coarse histogram $h_i$ of the relative coordinates of the remaining $n-1$ points,

$$h_i(k) \;\; = \;\; \# \left\{ q \neq p_i \; : \; (q - p_i) \in \text{bin}(k) \right\} \; .$$

This histogram is defined to be the *shape context* of $p_i$. The descriptor should be more sensitive to differences in nearby pixels. We therefore use a log-polar coordinate system (see Fig. 1(c)). All distances are measured in units of $\alpha$ where $\alpha$ is the median distance between the $n^2$ point pairs in the shape.

Note that the construction ensures that global translation or scaling of a shape will not affect the shape contexts. Since shape contexts are extremely rich descriptors, they are inherently tolerant to small perturbations of parts of the shape. While we have no theoretical guarantees here, robustness to small affine transformations, occlusions and presence of outliers is evaluated experimentally in [2]. Modifications to the shape context definition that provide for complete rotation invariance can alos be provided [2].

Consider a point $p_i$ on the first shape and a point $q_j$ on the second shape. Let $C_{ij} = C(p_i, q_j)$ denote the cost of matching these two points. As shape contexts are

distributions represented as histograms, it is natural[1] to use the $\chi^2$ test statistic:

$$C_{ij} = \frac{1}{2} \sum_{k=1}^{K} \frac{[h_i(k) - h_j(k)]^2}{h_i(k) + h_j(k)}$$

where $h_i(k)$ and $h_j(k)$ denote the $K$-bin normalized histogram at $p_i$ and $q_j$.

The cost $C_{ij}$ for matching points can include an additional term based on the local *appearance similarity* at points $p_i$ and $q_j$. This is particularly useful when we are comparing shapes derived from gray-level images instead of line drawings. For example, one can add a cost based on color or texture similarity, SSD between small gray-scale patches, distance between vectors of filter outputs, similarity of tangent angles, and so on. The choice of this appearance similarity term is application dependent, and is driven by the necessary invariance and robustness requirements, e.g. varying lighting conditions make reliance on gray-scale brightness values risky.

Given the set of costs $C_{ij}$ between all pairs of points $i$ on the first shape and $j$ on the second shape we want to minimize the total cost of matching subject to the constraint that the matching be one-to-one. This is an instance of the square assignment (or weighted bipartite matching) problem, which can be solved in $O(N^3)$ time using the Hungarian method. In our experiments, we use the more efficient algorithm of [10]. The input is a square cost matrix with entries $C_{ij}$. The result is a permutation $\pi(i)$ such that the sum $\sum_i C_{i,\pi(i)}$ is minimized.

When the number of samples on two shapes is not equal, the cost matrix can be made square by adding "dummy" nodes to each point set with a constant matching cost of $\epsilon_d$. The same technique may also be used even when the sample numbers are equal to allow for robust handling of outliers. In this case, a point will be matched to a "dummy" whenever there is no real match available at smaller cost than $\epsilon_d$. Thus, $\epsilon_d$ can be regarded as a threshold parameter for outlier detection.

Given a set of sample point correspondences between two shapes, one can proceed to estimate a transformation that maps one shape into the other. For this purpose there are several options; perhaps most common is the affine model. In this work, we use the thin plate spline (TPS) model, which is commonly used for representing flexible coordinate transformations [17, 6]. Bookstein [4], for example, found it to be highly effective for modeling changes in biological forms. The thin plate spline is the 2D generalization of the cubic spline, and in its regularized form, includes affine transformations as a limiting case. Our complete matching algorithm is obtained by alternating between the steps of recovering correspondences and estimating transformations. We usually employ a fixed number of iterations, typically three in large scale experiments, but more refined schemes are possible. However, experimental experiences show that the algorithmic performance is independent of the details. More details may be found in [2].

As far as we are aware, the shape context descriptor and its use for matching 2D shapes is novel. A related idea in past work is that due to Johnson and Hebert [9] in their work on range images. They introduced a representation for matching dense clouds of oriented 3D points called the "spin image". A spin image is a 2D histogram formed by spinning a plane around a normal vector on the surface of the object and counting the points that fall inside bins in the plane.

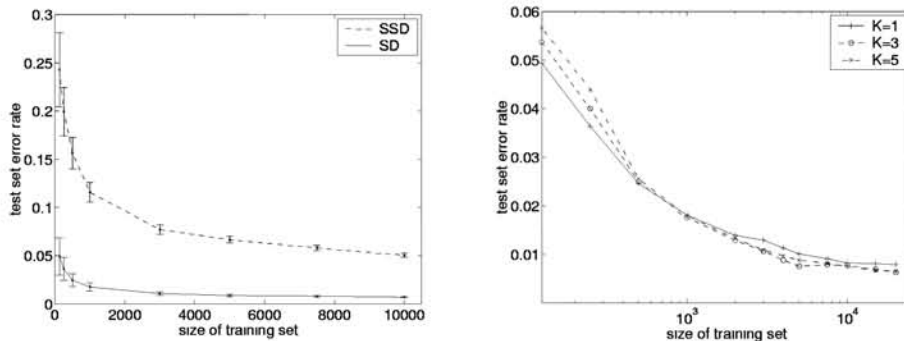

Figure 2: Handwritten digit recognition on the MNIST dataset. Left: Test set errors of a 1-NN classifier using SSD and Shape Distance (SD) measures. Right: Detail of performance curve for Shape Distance, including results with training set sizes of 15,000 and 20,000. Results are shown on a semilog-$x$ scale for $K = 1, 3, 5$ nearest neighbors.

## 3   Classification using Shape Context matching

Matching shapes enables us to define distances between shapes; given such a distance measure a straightforward strategy for recognition is to use a $K$-NN classifier. In the following two case studies we used 100 point samples selected from the Canny edges of each image. We employed a regularized TPS transformation model and used 3 iterations of shape context matching and TPS re-estimation. After matching, we estimated shape distances as the weighted sum of three terms: shape context distance, image appearance distance and bending energy.

We measure shape context distance between shapes $\mathcal{P}$ and $\mathcal{Q}$ as the symmetric sum of shape context matching costs over best matching points, i.e.

$$D_{\mathrm{sc}}(\mathcal{P}, \mathcal{Q}) = \frac{1}{n} \sum_{p \in \mathcal{P}} \arg \min_{q \in \mathcal{Q}} C(p, T(q)) + \frac{1}{m} \sum_{q \in \mathcal{Q}} \arg \min_{p \in \mathcal{P}} C(p, T(q)) \quad (1)$$

where $T(\cdot)$ denotes the estimated TPS shape transformation. We use a term $D_{\mathrm{ac}}(\mathcal{P}, \mathcal{Q})$ for appearance cost, defined as the sum of squared brightness differences in Gaussian windows around corresponding image points. This score is computed *after* the thin plate spline transformation $T$ has been applied to best warp the images into alignment. The third term $D_{\mathrm{be}}(\mathcal{P}, \mathcal{Q})$ corresponds to the 'amount' of transformation necessary to align the shapes. In the TPS case the *bending energy* is a natural measure (see [4, 2]).

**Case study 1: Digit recognition**   Here we present results on the MNIST dataset of handwritten digits, which consists of 60,000 training and 10,000 test digits [12].

Nearest neighbor classifiers have the property that as the number of examples $n$ in the training set goes to infinity, the 1-NN error converges to a value $\leq 2E^*$, where $E^*$ is the Bayes Risk (for $K$-NN, $K \to \infty$ and $K/n \to 0$, the error $\to E^*$). However, what matters in practice is the performance for small $n$, and this gives us a way to compare different similarity/distance measures. In Fig. 2, our shape distance is compared to SSD (sum of squared differences between pixel brightness values).

On the MNIST dataset nearly 30 algorithms have been compared (http://www.research.att.com/~yann/exdb/mnist/index.html). The lowest test set error rate published at this time is 0.7% for a boosted LeNet-4 with a training set of size

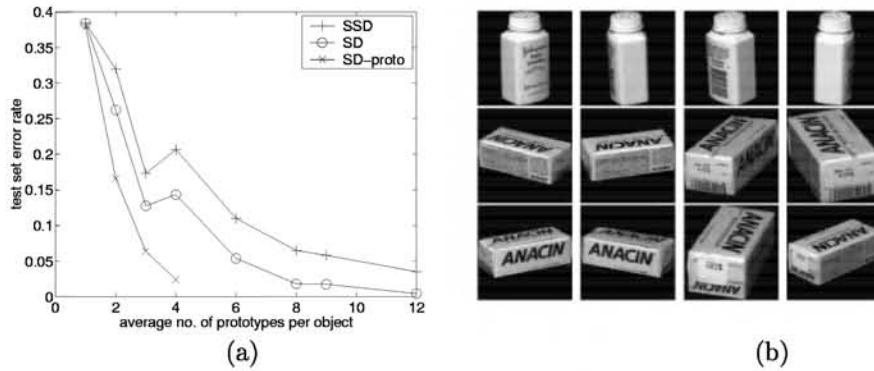

|     |     |
| :-: | :-: |
| (a) | (b) |

Figure 3: 3D object recognition. (a) Comparison of test set error for SSD, Shape Distance (SD), and Shape Distance with $K$-medoid prototypes (SD-proto) vs. number of prototype views. For SSD and SD, we varied the number of prototypes uniformly for all objects. For SD-proto, the number of prototypes per object depended on the within-object variation as well as the between-object similarity. (b) $K$-medoid prototype views for two different examples, using an average of 4 prototypes per object.

$60,000 \times 10$ synthetic distortions per training digit. Our error rate using 20,000 training examples and 3-NN is 0.63%.

**Case study 2: 3D object recognition** Our next experiment involves the 20 common household objects from the COIL-20 database [13]. We prepared our training sets by selecting a number of equally spaced views for each object and using the remaining views for testing. The matching algorithm is exactly the same as for digits. Fig. 3(a) shows the performance using 1-NN on the weighted shape distance compared to a straightforward sum of squared differences (SSD). SSD performs very well on this easy database due to the lack of variation in lighting [8].

Since the objects in the COIL-20 database have differing variability with respect to viewing angle, it is natural to ask whether prototypes can be allocated more efficiently. We have developed a novel *editing* algorithm based on shape distance and $K$-medoid clustering. $K$-medoids can be seen as a variant of $K$-means that restricts prototype positions to data points. First a matrix of pairwise similarities between all possible prototypes is computed. For a given number of $K$ prototypes the $K$-medoid algorithm then iterates two steps: (i) For a given assignment of points to (abstract) clusters a prototype is selected by minimizing the average distance of the prototype to all elements in the cluster, and (ii) given the set of prototypes, points are then reassigned to clusters according to the nearest prototype. The number of prototypes is selected by a greedy splitting strategy starting from one prototype per category. We choose the cluster to split based on the associated overall misclassification error. This continues until the overall misclassification error has dropped below a criterion level.

The editing algorithm is illustrated in Fig. 3(b). As seen, more prototypes are allocated to categories with high within class variability. The curve marked SD-proto in Fig. 3 shows the improved classification performance using this prototype selection strategy instead of equally-spaced views. Note that we obtain a 2.4% error rate with an average of only 4 two-dimensional views for each three-dimensional object, thanks to the flexibility provided by the matching algorithm.

# 4 Conclusion

We have presented a new approach to computing shape similarity and correspondences based on the shape context descriptor. Appealing features of our approach are its simplicity and robustness. The standard invariances are built in for free, and as a consequence we developed a classifier that is highly effective even when only a small number of training examples are available.

**Acknowledgments**  This research is supported by (ARO) DAAH04-96-1-0341, the Digital Library Grant IRI-9411334, an NSF graduate Fellowship for S.B and the German Research Foundation (DFG) by Emmy Noether grant PU-165/1.

## Footnotes

[1]Alternatives include Bickel's generalization of the Kolmogorov-Smirnov test for 2D distributions [3], which does not require binning.

# References

[1] Y. Amit, D. Geman, and K. Wilder. Joint induction of shape features and tree classifiers. *IEEE Trans. PAMI*, 19(11):1300–1305, November 1997.

[2] S. Belongie, J. Malik, and J. Puzicha. Shape matching and object recognition using shape contexts. Technical report, UC Berkeley, January 2001.

[3] P. J. Bickel. A distribution free version of the Smirnov two-sample test in the multivariate case. *Annals of Mathematical Statistics*, 40:1–23, 1969.

[4] F. L. Bookstein. Principal warps: thin-plate splines and decomposition of deformations. *IEEE Trans. PAMI*, 11(6):567–585, June 1989.

[5] C. Burges and B. Schölkopf. Improving the accuracy and speed of support vector machines. In *NIPS*, pages 375–381, 1997.

[6] H. Chui and A. Rangarajan. A new algorithm for non-rigid point matching. In *CVPR*, volume 2, pages 44–51, June 2000.

[7] S. Gold, A. Rangarajan, C-P. Lu, S. Pappu, and E. Mjolsness. New algorithms for 2D and 3D point matching: pose estimation and correspondence. *Pattern Recognition*, 31(8), 1998.

[8] D.P. Huttenlocher, R. Lilien, and C. Olson. View-based recognition using an eigenspace approximation to the Hausdorff measure. *PAMI*, 21(9):951–955, Sept. 1999.

[9] Andrew E. Johnson and Martial Hebert. Recognizing objects by matching oriented points. In *CVPR*, pages 684–689, 1997.

[10] R. Jonker and A. Volgenant. A shortest augmenting path algorithm for dense and sparse linear assignment problems. *Computing*, 38:325–340, 1987.

[11] M. Lades, C.C. Vorbrüggen, J. Buhmann, J. Lange, C. von der Malsburg, R.P. Wurtz, and W. Konen. Distortion invariant object recognition in the dynamic link architecture. *IEEE Trans. Computers*, 42(3):300–311, March 1993.

[12] Y. LeCun, L. Bottou, Y. Bengio, and P. Haffner. Gradient-based learning applied to document recognition. *Proceedings of the IEEE*, 86(11):2278–2324, November 1998.

[13] H. Murase and S.K. Nayar. Visual learning and recognition of 3-D objects from appearance. *Int. Journal of Computer Vision*, 14(1):5–24, Jan. 1995.

[14] M. Oren, C. Papageorgiou, P. Sinha, E. Osuna, and T. Poggio. Pedestrian detection using wavelet templates. In *CVPR*, pages 193–199, Puerto Rico, June 1997.

[15] M. Turk and A.P. Pentland. Eigenfaces for recognition. *J. Cognitive Neuroscience*, 3(1):71–96, 1991.

[16] R. C. Veltkamp and M. Hagedoorn. State of the art in shape matching. Technical Report UU-CS-1999-27, Utrecht, 1999.

[17] G. Wahba. *Spline Models for Observational Data*. SIAM, 1990.
